# The Role of Activity in Synaptic Competition at the Neuromuscular Junction

**Samuel R. H. Joseph**
Centre for Cognitive Science
Edinburgh University
Edinburgh, U.K.
email: sam@cns.ed.ac.uk

**David J. Willshaw**
Centre for Cognitive Science
Edinburgh University
Edinburgh, U.K.
email: david@cns.ed.ac.uk

## Abstract

An extended version of the dual constraint model of motor end-plate morphogenesis is presented that includes activity dependent and independent competition. It is supported by a wide range of recent neurophysiological evidence that indicates a strong relationship between synaptic efficacy and survival. The computational model is justified at the molecular level and its predictions match the developmental and regenerative behaviour of real synapses.

## 1 INTRODUCTION

The neuromuscular junction (NMJ) of mammalian skeletal muscle is one of the most extensively studied areas of the nervous system. One aspect of its development that it shares with many other parts of the nervous system is its achievement of single innervation, one axon terminal connecting to one muscle fibre, after an initial state of polyinnervation. The presence of electrical activity is associated with this transition, but the exact relationship is far from clear. Understanding how activity interacts with the morphogenesis of neural systems could provide us with insights into methods for constructing artificial neural networks. With that in mind, this paper examines how some of the conflicting ideas about the development of neuromuscular connections can be resolved.

## 2 EXPERIMENTAL FINDINGS

The extent to which a muscle is innervated can be expressed in terms of the motor unit size - the number of fibres contacted by a given motor axon. Following removal of some motor axons at birth, the average size of the remaining motor units after withdrawal of polyinnervation is larger than normal (Fladby & Jansen, 1987). This strongly suggests that individual motor axons successfully innervate more fibres as a result of the absence of their neighbours. It is appealing to interpret this as a competitive process where terminals from different axons compete for the same muscle endplate. Since each terminal is made up of a number of synapses the process can be viewed as the co-existence of synapses from the same terminal and the elimination of synapses from different terminals on the same endplate.

### 2.1 THE EFFECTS OF ELECTRICAL ACTIVITY

There is a strong activity dependent component to synapse elimination. Paralysis or stimulation of selected motor units appears to favour the more active motor terminals (Colman & Lichtman, 1992), while inactive axon terminals tend to coexist. Recent work also shows that active synaptic sites can destabilise inactive synapses in their vicinity (Balice-Gordon & Lichtman, 1994). These findings support the idea that more active terminals have a competitive advantage over their inactive fellows, and that this competition takes place at a synaptic level.

Activity independent competition has been demonstrated in the rat lumbrical muscle (Ribchester, 1993). This muscle is innervated by the sural and the lateral plantar nerves. If the sural nerve is damaged the lateral plantar nerve will expand its territory to the extent that it innervates the entire muscle. On subsequent reinnervation the regenerating sural nerve may displace some of the lateral plantar nerve terminals. If the muscle is paralysed during reinnervation more lateral plantar nerve terminals are displaced than in the normal case, indicating that competition between inactive terminals does take place, and that paralysis can give an advantage to some terminals.

## 3 MODELS AND MECHANISMS

If the nerve terminals are competing with each other for dominance of motor endplates, what is the mechanism behind it? As mentioned above, activity is thought to play an important role in affecting the competitive chances of a terminal, but in most models the terminals compete for some kind of trophic resource (Gouze et al., 1983; Willshaw, 1981). It is possible to create models that use competition for either a postsynaptic (endplate) resource or a presynaptic (motor axon) resource. Both types of model have advantages and disadvantages, which leads naturally to the possibility of combining the two into a single model.

### 3.1 BENNET AND ROBINSON'S DUAL CONSTRAINT MODEL

The dual constraint model (DCM) (Bennet & Robinson, 1989), as extended by Rasmussen & Willshaw (1993), is based on a reversible reaction between molecules from a presynaptic resource $A$ and a postsynaptic resource $B$. This reaction takes place in the synaptic cleft and produces a binding complex $C$ which is essential for

the terminal's survival. Each motor axon and muscle fibre has a limited amount of their particular resource and the size of each terminal is proportional to the amount of the binding complex at that terminal. The model achieves single innervation and a perturbation analysis performed by Rasmussen & Willshaw (1993) showed that this single innervation state is stable. However, for the DCM to function the forward rate of the reaction had to be made proportional to the size of the terminal, which was difficult to justify other than suggesting it was related to electrical activity.

## 3.2  SELECTIVE MECHANISMS

While the synapses in the surviving presynaptic terminal are allowed to coexist, synapses from other axons are eliminated. How do synapses make a distinction between synapses in their own terminal and those in others? There are two possibilities: (i) Synchronous transmitter release in the synaptic boutons of a motor neuron could distinguish synapses, allowing them to compete as cartels rather than individuals (Colman & Lichtman, 1992). (ii) The synapses could be employing selective recognition mechanisms, e.g the 'induced-fit' model (Ribchester & Barry, 1994).

A selective mechanism implies that all the synapses of a given motor neuron can be identified by a molecular substrate. In the induced-fit model each motor neuron is associated with a specific isoform of a cellular adhesion molecule (CAM); the synapses compete by attempting to induce all the CAMs on the endplate into the conformation associated with their neuron. This kind of model can be used to account for much of the developmental and regenerative processes of the NMJ. However, it has difficulty explaining Balice-Gordon & Lichtman's (1994) focal blockade experiments which show competition between synapses distinguished only by the presence of activity. If, instead, activity is responsible for the distinction of friend from foe, how can competition take place at the terminal level when activity is not present? Could we resolve this dilemma by extending the dual constraint model?

# 4   EXTENDING THE DUAL CONSTRAINT MODEL

Tentative suggestions can be made for the identity of the 'mystery molecules' in the DCM. According to McMahan (1990) a protein called agrin is synthesised in the cell bodies of motor neurons and transported down their axons to the muscle. When this protein binds to the surface of the developing muscle, it causes acetylcholine receptors (AChRs), and other components of the postsynaptic apparatus, to aggregate on the myotube surface in the vicinity of the activated agrin.

Other work (Wallace, 1988) has provided insights into the mechanism used by agrin to cause the aggregation of the postsynaptic apparatus. Initially, AChR aggregates, or 'speckles', are free to diffuse laterally in the myotube plasma membrane (Axelrod et al., 1976). When agrin binds to an agrin-specific receptor, AChR speckles in the immediate vicinity of the agrin-receptor complex are immobilised. As more speckles are trapped larger patches are formed, until a steady state is reached. Such a patch will remain so long as agrin is bound to its receptor and $Ca^{++}$ and energy supplies are available.

Following AChR activation by acetylcholine, $Ca^{++}$ enters the postsynaptic cell. Since $Ca^{++}$ is required for both the formation and maintenance of AChR aggregates,

a feedback loop is possible whereby the bigger a patch is the more $Ca^{++}$ it will have available when the receptors are activated. Crucially, depolarisation of non-junctional regions blocks AChR expression (Andreose et al., 1995) and it is AChR activation at the NMJ that causes depolarisation of the postsynaptic cell. So it seems that agrin is a candidate for molecule $A$, but what about $B$ or $C$? It is tempting to posit AChR as molecule $B$ since it is the critical postsynaptic resource. However, since agrin does not bind directly to the acetylcholine receptor, a different sort of reaction is required.

## 4.1   A DIFFERENT SORT OF REACTION

If AChR is molecule $B$, and one agrin molecule can attract at least 160 AChRs (Nitkin et al., 1987) the simple reversible reaction of the DCM is ruled out. Alternatively, AChR could exist in either free, $B_f$, or bound, $B_b$ states, being converted through the mediation of $A$. $B_b$ would now play the role of $C$ in the DCM. It is possible to devise a rate equation for the change in the number of receptors at a nerve terminal over time:

$$\frac{dB_b}{dt} = \alpha A B_f - \beta B_b \tag{1}$$

where $\alpha$ and $\beta$ are rate constants. The increase in bound AChR over time is proportional to the amount of agrin at a junction and the number of free receptors in the endplate area, while the decrease is proportional to the amount of bound AChRs. The rate equation (1) can be used as the basis of an extended DCM if four other factors are considered: (i) Agrin stays active as receptors accumulate, so the conservation equations for $A$ and $B$ are:

$$A_0 = A_n + \sum_{j=1}^{M} A_{nj} \quad B_0 = B_{mf} + \sum_{i=1}^{N} B_{imb} \tag{2}$$

where the subscript 0 indicates the fixed resource available to each muscle or neuron, the lettered subscripts indicate the amount of that substance that is present in the neuron $n$, muscle fibre $m$ and terminal $nm$, and there are N motor neurons and M muscle fibres. (ii) The size of a terminal is proportional to the number of bound AChRs, so if we assume the anterograde flow is evenly divided between the $\nu_n$ terminals of neuron $n$, the transport equation for agrin is:

$$\frac{dA_{nm}}{dt} = \lambda \frac{A_n}{\nu_n} - \delta \frac{A_{nm}}{B_{nmb}} \tag{3}$$

where $\lambda$ and $\delta$ are transport rate constants and the retrograde flow is assumed proportional to the amount of agrin at the terminal and inversely proportional to the size of the terminal. (iii) AChRs are free to diffuse laterally across the surface of the muscle, so the forward reaction rate will be related to the probability of an AChR speckle intersecting a terminal, which is itself proportional to the terminal diameter. (iv) The influx of $Ca^{++}$ through AChRs on the surface of the endplate will also affect the forward reaction rate in proportion to the area of the terminal. Taking $B_b$ to be proportional to the volume of the postsynaptic apparatus, these last two terms are proportional to $B_b^{1/3}$ and $B_b^{2/3}$ respectively. This gives the final rate equation:

$$\frac{dB_{nmb}}{dt} = \alpha A_{nm} B_{mf} B_{nmb}^{1/3} B_{nmb}^{2/3} - \beta B_{nmb} = \alpha A_{nm} B_{mf} B_{nmb} - \beta B_{nmb} \tag{4}$$

Equations (3) and (4) are similar to those in the original DCM, only now we have been able to justify the dependence of the forward reaction rate on the size of the terminal, $B_{nmb}$. We can also resolve the distinction paradox, as follows.

## 4.2   RESOLVING THE DISTINCTION PARADOX

In terms of distinguishing between synapses it seems plausible that concurrently active synapses (i.e. those belonging to the same neuron) will protect themselves from the negative effects of depolarisation. In paralysed systems, synapses will benefit from the AChR accumulating affects of the agrin molecules in those synapses nearby (i.e. those in the same terminal). It was suggested (Jennings, 1994) that competition between synapses of the same terminal was seen after focal blockade because active AChRs help stabilise the receptors around them and suppress those further away. This fits in with the stabilisation role of $Ca^{++}$ in this model and the suppressive effects of depolarisation, as well as the physical range of these effects during 'heterosynaptic suppression' (Lo & Poo, 1991). It seems that Jenning's mechanism, although originally speculative, is actually quite a plausible explanation and one that fits in well with the extended DCM. The critical effect in the XDCM is that if the system is paralysed during development there is a change in the dependency of the forward reaction rate on the size of an individual terminal. This gives the reinnervating terminals a small initial advantage due to their more competitive diameter/volume ratios. As we shall see in the next section, this allows us to demonstrate activity independent competition.

## 5   SIMULATING THE EXTENDED DCM

In terms of achieving single innervation the extended DCM performs just as well as the original, and when subjected to the same perturbation analysis it has been demonstrated to be stable. Simulating a number of systems with as many muscle fibres and motor neurons as found in real muscles allowed a direct comparison of model findings with experimental data (figure 1).

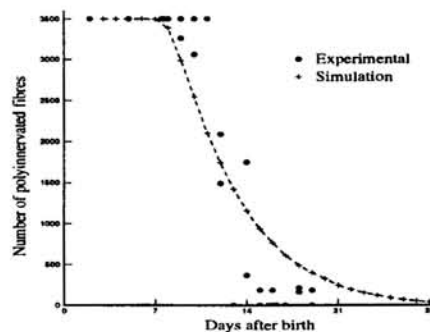

Figure 1: Elimination of Polyinnervation in Rat soleus muscle and Simulation

Figure 2 shows nerve dominance histograms of reinnervation in both the rat lumbrical muscle and its extended DCM simulation. Both compare the results produced when the system is paralysed from the outset of reinnervation (removal of $B_{nmb}^{2/3}$

term from equation (4)) with the normal situation. Note that in both the simulation and the experiment the percentage of fibres singly innervated by the reinnervating sural nerve is increased in the paralysis case. Inactive sural nerve terminals are displacing more inactive lateral plantar nerve terminals (activity independent competition). They can achieve this because during paralysis the terminals with the largest diameters capture more receptors, while the terminals with the largest volumes lose more agrin; so small reinnervating terminals do a little better. However, if activity is present the receptors are captured in proportion to a terminal's volume, so there's no advantage to a small terminal's larger diameter/volume ratio.

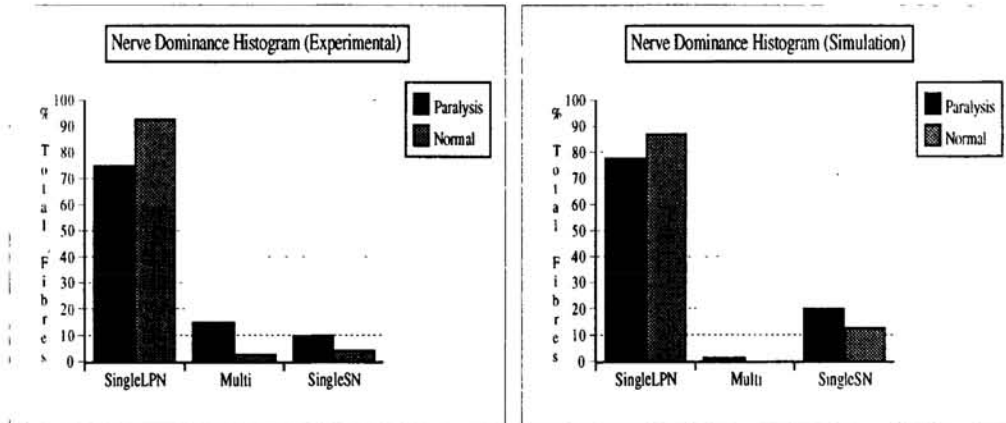

Figure 2: Types of Innervation by Lateral Plantar and Sural Nerves

## 6 DISCUSSION

The extensions to the DCM outlined here demonstrate both activity dependent and independent competition and provide greater biochemical plausibility. However this is still only a phenomenological demonstration and further experimental work is required to ascertain its validity. There is a need for illumination concerning the specific chemical mechanisms that underlie agrin's aggregational effects and the roles that both $Ca^{++}$ and depolarisation play in junctional dynamics. An important connection made here is one between synaptic efficiency and junctional survival. $Ca^{++}$ and NO have both been implicated in Hebbian mechanisms (Bliss & Collingridge, 1993) and perhaps some of the principles uncovered here may be applicable to neuroneuronic synapses. This work should be followed up with a direct model of synaptic interaction at the NMJ that includes the presynaptic effects of depolarisation, allowing the efficacy of the synapse to be related to its biochemistry; an important step forward in our understanding of nervous system plasticity. Relating changes in synaptic efficiency to neural morphogenesis may also give insights into the construction of artificial neural networks.

### Acknowledgements

We are grateful to Michael Joseph and Bruce Graham for critical reading of the manuscript and to the M.R.C. for funding this work.

## References

Andreose J. S., Fumagalli G. & Lømo T. (1995) Number of junctional acetylcholine receptors: control by neural and muscular influences in the rat. *Journal of Physiology* **483.2**:397-406.

Axelrod D., Ravdin P., Koppel D. E., Schlessinger J., Webb W. W., Elson E. L. & Podleski T. R. (1976) Lateral motion of fluorescently labelled acetylcholine receptors in membranes of developing muscle fibers. *Proc. Natl. Acad. Sci. USA* **73**:4594-4598.

Balice-Gordon R. J. & Lichtman J. W. (1994) Long-term synapse loss induced by focal blockade of postsynaptic receptors. *Nature* **372**:519-524.

Bennett M. R. & Robinson J. (1989) Growth and elimination of nerve terminals during polyneuronal innervation of muscle cells: a trophic hypothesis. *Proc. Royal Soc. Lond. [Biol]* **235**:299-320.

Bliss T. V. P. & Collingridge G. L. (1993) A synaptic model of memory: long-term potentiation in the hippocampus. *Nature* **361**:31-39.

Colman H. & Lichtman J. W. (1992) 'Cartellian' competition at the neuromuscular junction. *Trends in Neuroscience* **15**, **6**:197-199.

Fladby T. & Jansen J. K. S. (1987) Postnatal loss of synaptic terminals in the partially denervated mouse soleus muscle. *Acta. Physiol. Scand* **129**:239-246.

Gouze J. L., Lasry J. M. & Changeux J. -P. (1983) Selective stabilization of muscle innervation during development: A mathematical model. *Biol Cybern.* **46**:207-215.

Jennings C. (1994) Death of a synapse. *Nature* **372**:498-499.

Lo Y. J. & Poo M. M. (1991) Activity-dependent synapse competition in vitro: heterosynaptic suppression of developing synapses. *Science* **254**:1019-1022.

McMahan U. J. (1990) The Agrin Hypothesis. *Cold Spring Harbour Symp. Quant. Biol.* **55**:407-419.

Nitkin R. M., Smith M. A., Magill C., Fallon J. R., Yao Y. -M. M., Wallace B. G. & McMahan U. J. (1987) Identification of agrin, a synaptic organising protein from Torpedo electric organ. *Journal Cell Biology* **105**:2471-2478.

Rasmussen C. E. & Willshaw D. J. (1993) Presynaptic and postsynatic competition in models for the development of neuromuscular connections. *B. Cyb.* **68**:409-419.

Ribchester R. R. (1993) Co-existence and elimination of convergent motor nerve terminals in reinnervated and paralysed adult rat skeletal muscle. *J. Phys.* **466**: 421-441.

Ribchester R. R. & Barry J. A. (1994) Spatial Versus Consumptive Competition at Polyneuronally Innervated Neuromuscular Junctions. *Exp. Physiology* **79**:465-494.

Wallace B. G. (1988) Regulation of agrin-induced acetylcholine receptor aggregation by $Ca^{++}$ and phorbol ester. *Journal of Cell Biol.* **107**:267-278.

Willshaw D. J. (1981) The establishment and the subsequent elimination of polyneuronal innervation of developing muscle: theoretical considerations. *Proc. Royal Soc. Lond.* **B212**: 233-252.
